# Dimensionality Reduction for Data in Multiple Feature Representations

**Yen-Yu Lin**[1,2]          **Tyng-Luh Liu**[1]          **Chiou-Shann Fuh**[2]

[1]Institute of Information Science, Academia Sinica, Taipei, Taiwan
{yylin, liutyng}@iis.sinica.edu.tw
[2]Department of CSIE, National Taiwan University, Taipei, Taiwan
fuh@csie.ntu.edu.tw

## Abstract

In solving complex visual learning tasks, adopting multiple descriptors to more precisely characterize the data has been a feasible way for improving performance. These representations are typically high dimensional and assume diverse forms. Thus finding a way to transform them into a unified space of lower dimension generally facilitates the underlying tasks, such as object recognition or clustering. We describe an approach that incorporates *multiple kernel learning* with *dimensionality reduction* (MKL-DR). While the proposed framework is flexible in simultaneously tackling data in various feature representations, the formulation itself is general in that it is established upon *graph embedding*. It follows that any dimensionality reduction techniques explainable by graph embedding can be generalized by our method to consider data in multiple feature representations.

## 1 Introduction

The fact that most visual learning problems deal with high dimensional data has made dimensionality reduction an inherent part of the current research. Besides having the potential for a more efficient approach, working with a new space of lower dimension often can gain the advantage of better analyzing the intrinsic structures in the data for various applications, *e.g.*, [3, 7]. However, despite the great applicability, the existing dimensionality reduction methods suffer from two main restrictions. First, many of them, especially the linear ones, require data to be represented in the form of feature vectors. The limitation may eventually reduce the effectiveness of the overall algorithms when the data of interest could be more precisely characterized in other forms, such as bag-of-features [1, 11] or high order tensors [19]. Second, there seems to be lacking a systematic way of integrating multiple image features for dimensionality reduction. When addressing applications that no single descriptor can appropriately depict the whole dataset, this shortcoming becomes even more evident. Alas, it is usually the case for addressing complex visual learning tasks [4].

Aiming to relax the two above-mentioned restrictions, we introduce an approach called MKL-DR that incorporates *multiple kernel learning* (MKL) into the training process of dimensionality reduction (DR) algorithms. Our approach is inspired by the work of Kim *et al.* [8], in which learning an optimal kernel over a given convex set of kernels is coupled with *kernel Fisher discriminant analysis* (KFDA), but their method only considers binary-class data. Without the restriction, MKL-DR manifests its flexibility in two aspects. First, it works with multiple *base kernels*, each of which is created based on a specific kind of visual feature, and combines these features in the domain of kernel matrices. Second, the formulation is illustrated with the framework of *graph embedding* [19], which presents a unified view for a large family of DR methods. Therefore the proposed MKL-DR is ready to generalize any DR methods if they are expressible by graph embedding. Note that these DR methods include supervised, semisupervised and unsupervised ones.

## 2 Related work

This section describes some of the key concepts used in the establishment of the proposed approach, including graph embedding and multiple kernel learning.

### 2.1 Graph embedding

Many dimensionality reduction methods focus on modeling the pairwise relationships among data, and utilize graph-based structures. In particular, the framework of graph embedding [19] provides a unified formulation for a set of DR algorithms. Let $\Omega = \{\mathbf{x}_i \in \mathbb{R}^d\}_{i=1}^N$ be the dataset. A DR scheme accounted for by graph embedding involves a complete graph $G$ whose vertices are over $\Omega$. An affinity matrix $W = [w_{ij}] \in \mathbb{R}^{N \times N}$ is used to record the edge weights that characterize the similarity relationships between training sample pairs. Then the optimal linear embedding $\mathbf{v}^* \in \mathbb{R}^d$ can be obtained by solving

$$\mathbf{v}^* = \underset{\substack{\mathbf{v}^\top XDX^\top \mathbf{v}=1, \text{ or} \\ \mathbf{v}^\top XL'X^\top \mathbf{v}=1}}{\arg\min} \quad \mathbf{v}^\top XLX^\top \mathbf{v}, \tag{1}$$

where $X = [\mathbf{x}_1 \ \mathbf{x}_2 \ \cdots \ \mathbf{x}_N]$ is the data matrix, and $L = \operatorname{diag}(W \cdot \mathbf{1}) - W$ is the graph Laplacian of $G$. Depending on the property of a problem, one of the two constraints in (1) will be used in the optimization. If the first constraint is chosen, a diagonal matrix $D = [d_{ij}] \in \mathbb{R}^{N \times N}$ is included for scale normalization. Otherwise another complete graph $G'$ over $\Omega$ is required for the second constraint, where $L'$ and $W' = [w'_{ij}] \in \mathbb{R}^{N \times N}$ are respectively the graph Laplacian and affinity matrix of $G'$. The meaning of (1) can be better understood with the following equivalent problem:

$$\min_{\mathbf{v}} \quad \sum_{i,j=1}^N ||\mathbf{v}^\top \mathbf{x}_i - \mathbf{v}^\top \mathbf{x}_j||^2 w_{ij} \tag{2}$$

$$\text{subject to} \quad \sum_{i=1}^N ||\mathbf{v}^\top \mathbf{x}_i||^2 d_{ii} = 1, \text{ or} \tag{3}$$

$$\sum_{i,j=1}^N ||\mathbf{v}^\top \mathbf{x}_i - \mathbf{v}^\top \mathbf{x}_j||^2 w'_{ij} = 1. \tag{4}$$

The constrained optimization problem (2) implies that pairwise distances or distances to the origin of projected data (in the form of $\mathbf{v}^\top \mathbf{x}$) are modeled by one or two graphs in the framework. By specifying $W$ and $D$ (or $W$ and $W'$), Yan *et al.* [19] show that a set of dimensionality reduction methods, such as PCA, LPP [7], LDA, and MFA [19] can be expressed by (1).

### 2.2 Multiple kernel learning

MKL refers to the process of learning a kernel machine with multiple kernel functions or kernel matrices. Recent research efforts on MKL, *e.g.*, [9, 14, 16] have shown that learning SVMs with multiple kernels not only increases the accuracy but also enhances the interpretability of the resulting classifier. Our MKL formulation is to find an optimal way to linearly combine the given kernels. Suppose we have a set of base kernel functions $\{k_m\}_{m=1}^M$ (or base kernel matrices $\{K_m\}_{m=1}^M$). An *ensemble kernel function* $k$ (or an *ensemble kernel matrix* $K$) is then defined by

$$k(\mathbf{x}_i, \mathbf{x}_j) = \sum_{m=1}^M \beta_m k_m(\mathbf{x}_i, \mathbf{x}_j), \quad \beta_m \geq 0, \tag{5}$$

$$K = \sum_{m=1}^M \beta_m K_m, \quad \beta_m \geq 0. \tag{6}$$

Consequently, the learned model from binary-class data $\{(\mathbf{x}_i, y_i \in \pm 1)\}$ will be of the form:

$$f(\mathbf{x}) = \sum_{i=1}^N \alpha_i y_i k(\mathbf{x}_i, \mathbf{x}) + b = \sum_{i=1}^N \alpha_i y_i \sum_{m=1}^M \beta_m k_m(\mathbf{x}_i, \mathbf{x}) + b. \tag{7}$$

Optimizing both the coefficients $\{\alpha_i\}_{i=1}^N$ and $\{\beta_m\}_{m=1}^M$ is one particular form of the MKL problems. Our approach leverages such an MKL optimization to yield more flexible dimensionality reduction schemes for data in different feature representations.

## 3 The MKL-DR framework

To establish the proposed method, we first discuss the construction of a set of base kernels from multiple features, and then explain how to integrate these kernels for dimensionality reduction. Finally, we design an optimization procedure to learn the projection for dimensionality reduction.

### 3.1 Kernel as a unified feature representation

Consider a dataset $\Omega$ of $N$ samples, and $M$ kinds of descriptors to characterize each sample. Let $\Omega = \{\mathbf{x}_i\}_{i=1}^N$, $\mathbf{x}_i = \{\mathbf{x}_{i,m} \in \mathcal{X}_m\}_{m=1}^M$, and $d_m : \mathcal{X}_m \times \mathcal{X}_m \to 0 \cup \mathbb{R}^+$ be the distance function for data representation under the $m$th descriptor. The domains resulting from distinct descriptors, *e.g.* feature vectors, histograms, or bags of features, are in general different. To eliminate these varieties in representation, we represent data under each descriptor as a kernel matrix. There are several ways to accomplish this goal, such as using RBF kernel for data in the form of vector, or *pyramid match kernel* [6] for data in the form of bag-of-features. We may also convert pairwise distances between data samples to a kernel matrix [18, 20]. By coupling each representation and its corresponding distance function, we obtain a set of $M$ *dissimilarity-based* kernel matrices $\{K_m\}_{m=1}^M$ with

$$K_m(i,j) = k_m(\mathbf{x}_i, \mathbf{x}_j) = \exp\left\{\left(-d_m^2(\mathbf{x}_{i,m}, \mathbf{x}_{j,m})/\sigma_m^2\right)\right\} \tag{8}$$

where $\sigma_m$ is a positive constant. As several well-designed descriptors and their associated distance functions have been introduced over the years, the use of dissimilarity-based kernel is convenient in solving visual learning tasks. Nonetheless, care must be taken in that the resulting $K_m$ is not guaranteed to be positive semidefinite. Zhang *et al.* [20] have suggested a solution to resolve this issue. It follows from (5) and (6) that determining a set of optimal ensemble coefficients $\{\beta_1, \beta_2, \ldots, \beta_M\}$ can be interpreted as finding appropriate weights for best fusing the $M$ feature representations.

### 3.2 The MKL-DR algorithm

Instead of designing a specific dimensionality reduction algorithm, we choose to describe MKL-DR upon graph embedding. This way we can derive a general framework: If a dimensionality reduction scheme is explained by graph embedding, then it will also be extendible by MKL-DR to handle data in multiple feature representations. In graph embedding (2), there are two possible types of constraints. For the ease of presentation, we discuss how to develop MKL-DR subject to constraint (4). However, the derivation can be analogously applied when using constraint (3).

It has been shown that a set of linear dimensionality reduction methods can be *kernelized* to nonlinear ones via kernel trick. The procedure of kernelization in MKL-DR is mostly accomplished in a similar way, but with the key difference in using multiple kernels $\{K_m\}_{m=1}^M$. Suppose the ensemble kernel $K$ in MKL-DR is generated by linearly combining the base kernels $\{K_m\}_{m=1}^M$ as in (6). Let $\phi : \mathcal{X} \to \mathcal{F}$ denote the feature mapping induced by $K$. Through $\phi$, the training data can be implicitly mapped to a high dimensional Hilbert space, i.e.,

$$\mathbf{x}_i \mapsto \phi(\mathbf{x}_i), \text{ for } i = 1, 2, ..., N. \tag{9}$$

By assuming the optimal projection $\mathbf{v}$ lies in the span of training data in the feature space, we have

$$\mathbf{v} = \sum_{n=1}^N \alpha_n \phi(\mathbf{x}_n). \tag{10}$$

To show that the underlying algorithm can be reformulated in the form of inner product and accomplished in the new feature space $\mathcal{F}$, we observe that plugging into (2) each mapped sample $\phi(\mathbf{x}_i)$ and projection $\mathbf{v}$ would appear exclusively in the form of $\mathbf{v}^T\phi(\mathbf{x}_i)$. Hence, it suffices to show that in MKL-DR, $\mathbf{v}^T\phi(\mathbf{x}_i)$ can be evaluated via the kernel trick:

$$\mathbf{v}^T\phi(\mathbf{x}_i) = \sum_{n=1}^N \sum_{m=1}^M \alpha_n \beta_m k_m(\mathbf{x}_n, \mathbf{x}_i) = \boldsymbol{\alpha}^T \mathbb{K}^{(i)} \boldsymbol{\beta} \quad \text{where} \tag{11}$$

$$\boldsymbol{\alpha} = \begin{bmatrix} \alpha_1 \\ \vdots \\ \alpha_N \end{bmatrix} \in \mathbb{R}^N, \boldsymbol{\beta} = \begin{bmatrix} \beta_1 \\ \vdots \\ \beta_M \end{bmatrix} \in \mathbb{R}^M, \mathbb{K}^{(i)} = \begin{bmatrix} K_1(1,i) & \cdots & K_M(1,i) \\ \vdots & \ddots & \vdots \\ K_1(N,i) & \cdots & K_M(N,i) \end{bmatrix} \in \mathbb{R}^{N \times M}.$$

With (2) and (11), we define the constrained optimization problem for 1-D MKL-DR as follows:

$$\min_{\boldsymbol{\alpha}, \boldsymbol{\beta}} \quad \sum_{i,j=1}^N ||\boldsymbol{\alpha}^T \mathbb{K}^{(i)} \boldsymbol{\beta} - \boldsymbol{\alpha}^T \mathbb{K}^{(j)} \boldsymbol{\beta}||^2 w_{ij} \tag{12}$$

$$\text{subject to} \quad \sum_{i,j=1}^N ||\boldsymbol{\alpha}^T \mathbb{K}^{(i)} \boldsymbol{\beta} - \boldsymbol{\alpha}^T \mathbb{K}^{(j)} \boldsymbol{\beta}||^2 w'_{ij} = 1, \tag{13}$$

$$\beta_m \geq 0, \ m = 1, 2, ..., M. \tag{14}$$

The additional constraints in (14) are included to ensure the the resulting kernel $K$ in MKL-DR is a non-negative combination of base kernels. We leave the details of how to solve (12) until the next section, where using MKL-DR for finding a multi-dimensional projection $V$ is considered.

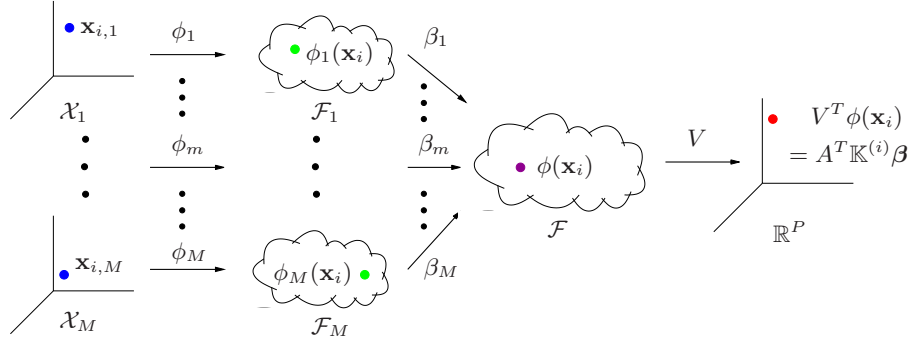

Figure 1: Four kinds of spaces in MKL-DR: the input space of each feature representation, the RKHS induced by each base kernel, the RKHS by the ensemble kernel, and the projected space.

### 3.3 Optimization

Observe from (11) that the one-dimensional projection $\mathbf{v}$ of MKL-DR is specified by a *sample coefficient vector* $\boldsymbol{\alpha}$ and a *kernel weight vector* $\boldsymbol{\beta}$. The two vectors respectively account for the relative importance among the samples and the base kernels. To generalize the formulation to uncover a multi-dimensional projection, we consider a set of $P$ sample coefficient vectors, denoted by

$$A = [\boldsymbol{\alpha}_1 \ \boldsymbol{\alpha}_2 \ \cdots \ \boldsymbol{\alpha}_P]. \tag{15}$$

With $A$ and $\boldsymbol{\beta}$, each 1-D projection $\mathbf{v}_i$ is determined by a specific sample coefficient vector $\boldsymbol{\alpha}_i$ and the (shared) kernel weight vector $\boldsymbol{\beta}$. The resulting projection $V = [\mathbf{v}_1 \ \mathbf{v}_2 \ \cdots \ \mathbf{v}_P]$ will map samples to a $P$-dimensional space. Analogous to the 1-D case, a projected sample $\mathbf{x}_i$ can be written as

$$V^\top \phi(\mathbf{x}_i) = A^\top \mathbb{K}^{(i)} \boldsymbol{\beta} \in \mathbb{R}^P. \tag{16}$$

The optimization problem (12) can now be extended to accommodate multi-dimensional projection:

$$\min_{A,\boldsymbol{\beta}} \quad \sum_{i,j=1}^{N} ||A^\top \mathbb{K}^{(i)} \boldsymbol{\beta} - A^\top \mathbb{K}^{(j)} \boldsymbol{\beta}||^2 w_{ij} \tag{17}$$

$$\text{subject to} \quad \sum_{i,j=1}^{N} ||A^\top \mathbb{K}^{(i)} \boldsymbol{\beta} - A^\top \mathbb{K}^{(j)} \boldsymbol{\beta}||^2 w'_{ij} = 1,$$

$$\beta_m \geq 0, \ m = 1, 2, ..., M.$$

In Figure 1, we give an illustration of the four kinds of spaces related to MKL-DR, including the input space of each feature representation, the RKHS induced by each base kernel and the ensemble kernel, and the projected Euclidean space.

Since direct optimization to (17) is difficult, we instead adopt an iterative, two-step strategy to alternately optimize $A$ and $\boldsymbol{\beta}$. At each iteration, one of $A$ and $\boldsymbol{\beta}$ is optimized while the other is fixed, and then the roles of $A$ and $\boldsymbol{\beta}$ are switched. Iterations are repeated until convergence or a maximum number of iterations is reached.

**On optimizing $A$:** By fixing $\boldsymbol{\beta}$, the optimization problem (17) is reduced to

$$\min_A \quad \text{trace}(A^\top S_W^{\boldsymbol{\beta}} A)$$
$$\text{subject to} \quad \text{trace}(A^\top S_{W'}^{\boldsymbol{\beta}} A) = 1 \tag{18}$$

where

$$S_W^{\boldsymbol{\beta}} = \sum_{i,j=1}^{N} w_{ij} (\mathbb{K}^{(i)} - \mathbb{K}^{(j)}) \boldsymbol{\beta}\boldsymbol{\beta}^\top (\mathbb{K}^{(i)} - \mathbb{K}^{(j)})^\top, \tag{19}$$

$$S_{W'}^{\boldsymbol{\beta}} = \sum_{i,j=1}^{N} w'_{ij} (\mathbb{K}^{(i)} - \mathbb{K}^{(j)}) \boldsymbol{\beta}\boldsymbol{\beta}^\top (\mathbb{K}^{(i)} - \mathbb{K}^{(j)})^\top. \tag{20}$$

The problem (18) is a *trace ratio* problem, i.e., $\min_A \text{trace}(A^\top S_W^{\boldsymbol{\beta}} A)/\text{trace}(A^\top S_{W'}^{\boldsymbol{\beta}} A)$. A closed-form solution can be obtained by transforming (18) into the corresponding *ratio trace* problem, i.e., $\min_A \text{trace}[(A^\top S_{W'}^{\boldsymbol{\beta}} A)^{-1}(A^\top S_W^{\boldsymbol{\beta}} A)]$. Consequently, the columns of the optimal $A^* = [\boldsymbol{\alpha}_1 \ \boldsymbol{\alpha}_2 \cdots \boldsymbol{\alpha}_P]$ are the eigenvectors corresponding to the first $P$ smallest eigenvalues in

$$S_W^{\boldsymbol{\beta}} \boldsymbol{\alpha} = \lambda S_{W'}^{\boldsymbol{\beta}} \boldsymbol{\alpha}. \tag{21}$$

---

**Algorithm 1**: *MKL-DR*

---

**Input** : A DR method specified by two affinity matrices $W$ and $W'$ (cf. (2));
          Various visual features expressed by base kernels $\{K_m\}_{m=1}^M$ (cf. (8));
**Output**: Sample coefficient vectors $A = [\boldsymbol{\alpha}_1 \; \boldsymbol{\alpha}_2 \cdots \boldsymbol{\alpha}_P]$;   Kernel weight vector $\boldsymbol{\beta}$;
Make an initial guess for $A$ or $\boldsymbol{\beta}$;
**for** $t \leftarrow 1, 2, \ldots, T$ **do**
  1. Compute $S_W^{\boldsymbol{\beta}}$ in (19) and $S_{W'}^{\boldsymbol{\beta}}$ in (20);
  2. $A$ is optimized by solving the generalized eigenvalue problem (21);
  3. Compute $S_W^A$ in (23) and $S_{W'}^A$ in (24);
  4. $\boldsymbol{\beta}$ is optimized by solving optimization problem (25) via semidefinite programming;
**return** $A$ and $\boldsymbol{\beta}$;

---

**On optimizing $\boldsymbol{\beta}$:**   By fixing $A$, the optimization problem (17) becomes

$$\min_{\boldsymbol{\beta}} \quad \boldsymbol{\beta}^\top S_W^A \boldsymbol{\beta}$$

$$\text{subject to} \quad \boldsymbol{\beta}^\top S_{W'}^A \boldsymbol{\beta} = 1 \text{ and } \boldsymbol{\beta} \geq \mathbf{0} \tag{22}$$

where

$$S_W^A = \sum_{i,j=1}^N w_{ij} (\mathbb{K}^{(i)} - \mathbb{K}^{(j)})^\top A A^\top (\mathbb{K}^{(i)} - \mathbb{K}^{(j)}), \tag{23}$$

$$S_{W'}^A = \sum_{i,j=1}^N w'_{ij} (\mathbb{K}^{(i)} - \mathbb{K}^{(j)})^\top A A^\top (\mathbb{K}^{(i)} - \mathbb{K}^{(j)}). \tag{24}$$

The additional constraints $\boldsymbol{\beta} \geq \mathbf{0}$ cause that the optimization to (22) can no longer be formulated as a generalized eigenvalue problem. Indeed it now becomes a nonconvex *quadratically constrained quadratic programming* (QCQP) problem, and is known to be very difficult to solve. We instead consider solving its convex relaxation by adding an auxiliary variable $B$ of size $M \times M$:

$$\min_{\boldsymbol{\beta}, B} \quad \text{trace}(S_W^A B) \tag{25}$$

$$\text{subject to} \quad \text{trace}(S_{W'}^A B) = 1, \tag{26}$$

$$\mathbf{e}_m^T \boldsymbol{\beta} \geq 0, \; m = 1, 2, ..., M, \tag{27}$$

$$\begin{bmatrix} 1 & \boldsymbol{\beta}^T \\ \boldsymbol{\beta} & B \end{bmatrix} \succeq 0, \tag{28}$$

where $\mathbf{e}_m$ in (27) is a column vector whose elements are 0 except that its $m$th element is 1, and the constraint in (28) means that the square matrix is positive semidefinite. The optimization problem (25) is an SDP relaxation of the nonconvex QCQP problem (22), and can be efficiently solved by semidefinite programming (SDP). One can verify the equivalence between the two optimization problems (22) and (25) by replacing the constraint (28) with $B = \boldsymbol{\beta}\boldsymbol{\beta}^T$. In view of that the constraint $B = \boldsymbol{\beta}\boldsymbol{\beta}^T$ is nonconvex, it is relaxed to $B \succeq \boldsymbol{\beta}\boldsymbol{\beta}^T$. Applying the Schur complement lemma, $B \succeq \boldsymbol{\beta}\boldsymbol{\beta}^T$ can be equivalently expressed by the constraint in (28). (Refer to [17] for further details.) Note that the numbers of constraints and variables in (25) are respectively linear and quadratic to $M$, the number of the adopted descriptors. In practice the value of $M$ is often small. ($M = 7$ in our experiments.) Thus like most of the other DR methods, the computational bottleneck of our approach is still in solving the generalized eigenvalue problems.

Listed in Algorithm 1, the procedure of MKL-DR requires an initial guess to either $A$ or $\boldsymbol{\beta}$ in the alternating optimization. We have tried two possibilities: 1) $\boldsymbol{\beta}$ is initialized by setting all of its elements as 1 to equally weight each base kernel; 2) $A$ is initialized by assuming $AA^\top = I$. In our empirical testing, the second initialization strategy gives more stable performances, and is thus adopted in the experiments. Pertaining to the convergence of the optimization procedure, since SDP relaxation has been used, the values of objective function are not guaranteed to monotonically decrease throughout the iterations. Still, the optimization procedures rapidly converge after only a few iterations in all our experiments.

**Novel sample embedding.**   Given a testing sample $\mathbf{z}$, it is projected to the learned space of lower dimension by

$$\mathbf{z} \mapsto A^T \mathbb{K}^{(\mathbf{z})} \boldsymbol{\beta}, \text{ where } \mathbb{K}^{(\mathbf{z})} \in \mathbb{R}^{N \times M} \text{ and } \mathbb{K}^{(\mathbf{z})}(n, m) = k_m(\mathbf{x}_n, \mathbf{z}). \tag{29}$$

## 4 Experimental results

To evaluate the effectiveness of MKL-DR, we test the technique with the supervised visual learning task of object category recognition. In the application, two (base) DR methods and a set of descriptors are properly chosen to serve as the input to MKL-DR.

### 4.1 Dataset

The Caltech-101 image dataset [4] consists of 101 object categories and one additional class of background images. The total number of categories is 102, and each category contains roughly 40 to 800 images. Although each target object often appears in the central region of an image, the large class number and substantial intraclass variations still make the dataset very challenging. Still, the dataset provides a good test bed to demonstrate the advantage of using multiple image descriptors for complex recognition tasks. Since the images in the dataset are not of the same size, we resize them to around 60,000 pixels, without changing their aspect ratio.

To implement MKL-DR for recognition, we need to select some proper graph-based DR method to be generalized and a set of image descriptors, and then derive (in our case) a pair of affinity matrices and a set of base kernels. The details are described as follows.

### 4.2 Image descriptors

For the Caltech-101 dataset, we consider seven kinds of image descriptors that result in the seven base kernels (denoted below in bold and in abbreviation):

**GB-1/GB-2**: From a given image, we randomly sample 300 edge pixels, and apply *geometric blur* descriptor [1] to them. With these image features, we adopt the distance function, as is suggested in equation (2) of the work by Zhang *et al.* [20], to obtain the two dissimilarity-based kernels, each of which is constructed with a specific descriptor radius.

**SIFT-Dist**: The base kernel is analogously constructed as in **GB-2**, except now the *SIFT* descriptor [11] is used to extract features.

**SIFT-Grid**: We apply SIFT with three different scales to an evenly sampled grid of each image, and use $k$-means clustering to generate *visual words* from the resulting local features of all images. Each image can then be represented by a histogram over the visual words. The $\chi^2$ distance is used to derive this base kernel via (8).

**C2-SWP/C2-ML**: Biologically inspired features are also considered here. Specifically, both the *C2* features derived by Serre *et al.* [15] and by Mutch and Lowe [13] have been chosen. For each of the two kinds of C2 features, an RBF kernel is respectively constructed.

**PHOG**: We adopt the *PHOG* descriptor [2] to capture image features, and limit the pyramid level up to 2. Together with $\chi^2$ distance, the base kernel is established.

### 4.3 Dimensionality reduction methods

We consider two supervised DR schemes, namely, *linear discriminant analysis* (LDA) and *local discriminant embedding* (LDE) [3], and show how MKL-DR can generalize them. Both LDA and LDE perform discriminant learning on a fully labeled dataset $\Omega = \{(\mathbf{x}_i, y_i)\}_{i=1}^{N}$, but make different assumptions about data distribution: LDA assumes data of each class can be modeled by a Gaussian, while LDE assumes they spread as a submanifold. Each of the two methods can be specified by a pair of affinity matrices to fit the formulation of graph embedding (2), and the resulting MKL dimensionality reduction schemes are respectively termed as *MKL-LDA* and *MKL-LDE*.

**Affinity matrices for LDA:** The two affinity matrices $W = [w_{ij}]$ and $W' = [w'_{ij}]$ are defined as

$$w_{ij} = \begin{cases} 1/n_{y_i}, & \text{if } y_i = y_j, \\ 0, & \text{otherwise,} \end{cases} \quad \text{and} \quad w'_{ij} = \frac{1}{N}, \tag{30}$$

where $n_{y_i}$ is the number of data points with label $y_i$. See [19] for the derivation.

Table 1: Recognition rates (mean $\pm$ std %) for Caltech-101 dataset

| kernel(s) | method | number of classes | | method | number of classes | |
|---|---|---|---|---|---|---|
| | | 102 | 101 | | 102 | 101 |
| GB-1 | | $57.3 \pm 2.5$ | $57.7 \pm 0.7$ | | $57.1 \pm 1.4$ | $57.7 \pm 0.8$ |
| GB-2 | | $60.0 \pm 1.5$ | $60.6 \pm 1.5$ | | $60.9 \pm 1.4$ | $61.3 \pm 2.1$ |
| SIFT-Dist | | $53.0 \pm 1.4$ | $53.2 \pm 0.8$ | | $54.2 \pm 0.5$ | $54.6 \pm 1.5$ |
| SIFT-Grid | KFD | $48.8 \pm 1.9$ | $49.6 \pm 0.7$ | KLDE | $49.5 \pm 1.3$ | $50.1 \pm 0.3$ |
| C2-SWP | | $30.3 \pm 1.2$ | $30.7 \pm 1.5$ | | $31.1 \pm 1.5$ | $31.3 \pm 0.7$ |
| C2-ML | | $46.0 \pm 0.6$ | $46.8 \pm 0.9$ | | $45.8 \pm 0.2$ | $46.7 \pm 1.5$ |
| PHOG | | $41.8 \pm 0.6$ | $42.1 \pm 1.3$ | | $42.2 \pm 0.6$ | $42.6 \pm 1.3$ |
| - | KFD-*Voting* | $68.4 \pm 1.5$ | $68.9 \pm 0.3$ | KLDE-*Voting* | $68.4 \pm 1.4$ | $68.7 \pm 0.8$ |
| - | KFD-*SAMME* | $71.2 \pm 1.4$ | $72.1 \pm 0.7$ | KLDE-*SAMME* | $71.1 \pm 1.9$ | $71.3 \pm 1.2$ |
| All | MKL-LDA | $\mathbf{74.6 \pm 2.2}$ | $\mathbf{75.3 \pm 1.7}$ | MKL-LDE | $\mathbf{75.3 \pm 1.5}$ | $\mathbf{75.5 \pm 1.7}$ |

**Affinity matrices for LDE:** In LDE, not only the data labels but also the neighborhood relationships are simultaneously considered to construct the affinity matrices $W = [w_{ij}]$ and $W' = [w'_{ij}]$:

$$w_{ij} = \begin{cases} 1, & \text{if } y_i = y_j \wedge [i \in \mathcal{N}_k(j) \vee j \in \mathcal{N}_k(i)], \\ 0, & \text{otherwise}, \end{cases} \tag{31}$$

$$w'_{ij} = \begin{cases} 1, & \text{if } y_i \neq y_j \wedge [i \in \mathcal{N}_{k'}(j) \vee j \in \mathcal{N}_{k'}(i)], \\ 0, & \text{otherwise}. \end{cases} \tag{32}$$

where $i \in \mathcal{N}_k(j)$ means that sample $\mathbf{x}_i$ is one of the $k$ nearest neighbors for sample $\mathbf{x}_j$. The definitions of the affinity matrices are faithful to those in LDE [3]. However, since there are now multiple image descriptors, we need to construct an affinity matrix for data under each descriptor, and average the resulting affinity matrices from all the descriptors.

## 4.4 Quantitative results

Our experiment setting follows the one described by Zhang *et al.* [20]. From each of the 102 classes, we randomly pick 30 images where 15 of them are included for training and the other 15 images are used for testing. To avoid a biased implementation, we redo the whole process of learning by switching the roles of training and testing data. In addition, we also carry out the experiments without using the data from the the background class, since such setting is adopted in some of the related works, *e.g.*, [5]. Via MKL-DR, the data are projected to the learned space, and the recognition task is accomplished there by enforcing the *nearest-neighbor* rule.

Coupling the seven base kernels with the affinity matrices of LDA and LDE, we can respectively derive MKL-LDA and MKL-LDE using Algorithm 1. Their effectiveness is investigated by comparing with KFD (kernel Fisher discriminant) [12] and KLDE (kernel LDE) [3]. Since KFD considers only one base kernel at a time, we implement two strategies to take account of the classification outcomes from the seven resulting KFD classifiers. The first is named as KFD-*Voting*. It is constructed based on the voting result of the seven KFD classifiers. If there is any ambiguity in the voting result, the next nearest neighbor in each KFD classifier will be considered, and the process is continued until a decision on the class label can be made. The second is termed as KFD-*SAMME*. By viewing each KFD classifier as a multi-class weak learner, we boost them by *SAMME* [21], which is a multi-class generalization of AdaBoost. Analogously, we also have KLDE-*Voting* and KLDE-*SAMME*.

We report the mean recognition rates and the standard deviation in Table 1. First of all, MKL-LDA achieves a considerable performance gain of $14.6\%$ over the best recognition rate by the seven KFD classifiers. On the other hand, while KFD-*Voting* and KFD-*SAMME* try to combine the *separately* trained KFD classifiers, MKL-LDA *jointly* integrates the seven kernels into the learning process. The quantitative results show that MKL-LDA can make the most of fusing various feature descriptors, and improves the recognition rates from $68.4\%$ and $71.2\%$ to $74.6\%$. Similar improvements can also be observed for MKL-LDE.

The recognition rates $74.6\%$ in MKL-LDA and $75.3\%$ in MKL-LDE are favorably comparable to those by most of the existing approaches. In [6], Grauman and Darrell report a $50\%$ recognition

rate based on the pyramid matching kernel over data in bag-of-features representation. By combing shape and spatial information, *SVM-KNN* of Zhang *et al*. [20] achieves $59.05\%$. In Frome *et al*. [5], the accuracy rate derived by learning the local distances, one for each training sample, is $60.3\%$. Our related work [10] that performs adaptive feature fusing via locally combining kernel matrices has a recognition rate $59.8\%$. Multiple kernel learning is also used in Varma and Ray [18], and it can yield a top recognition rate of $87.82\%$ by integrating visual cues like shape and color.

## 5 Conclusions and discussions

The proposed MKL-DR technique is useful as it has the advantage of learning a unified space of low dimension for data in multiple feature representations. Our approach is general and applicable to most of the graph-based DR methods, and improves their performance. Such flexibilities allow one to make use of more prior knowledge for effectively analyzing a given dataset, including choosing a proper set of visual features to better characterize the data, and adopting a graph-based DR method to appropriately model the relationship among the data points. On the other hand, via integrating with a suitable DR scheme, MKL-DR can extend the multiple kernel learning framework to address not just the supervised learning problems but also the unsupervised and the semisupervised ones.

**Acknowledgements.** This work is supported in part by grants 95-2221-E-001-031-MY3 and 97-2221-E-001-019-MY3.

## References

[1] A. Berg, T. Berg, and J. Malik. Shape matching and object recognition using low distortion correspondences. In *CVPR*, 2005.

[2] A. Bosch, A. Zisserman, and X. Muñoz. Image classification using random forests and ferns. In *ICCV*, 2007.

[3] H.-T. Chen, H.-W. Chang, and T.-L. Liu. Local discriminant embedding and its variants. In *CVPR*, 2005.

[4] L. Fei-Fei, R. Fergus, and P. Perona. Learning generative visual models from few training examples: An incremental bayesian approach tested on 101 object categories. In *CVPR Workshop on Generative-Model Based Vision*, 2004.

[5] A. Frome, Y. Singer, and J. Malik. Image retrieval and classification using local distance functions. In *NIPS*, 2006.

[6] K. Grauman and T. Darrell. The pyramid match kernel: Efficient learning with sets of features. *JMLR*, 2007.

[7] X. He and P. Niyogi. Locality preserving projections. In *NIPS*, 2003.

[8] S.-J. Kim, A. Magnani, and S. Boyd. Optimal kernel selection in kernel fisher discriminant analysis. In *ICML*, 2006.

[9] G. Lanckriet, N. Cristianini, P. Bartlett, L. Ghaoui, and M. Jordan. Learning the kernel matrix with semidefinite programming. *JMLR*, 2004.

[10] Y.-Y. Lin, T.-L. Liu, and C.-S. Fuh. Local ensemble kernel learning for object category recognition. In *CVPR*, 2007.

[11] D. Lowe. Distinctive image features from scale-invariant keypoints. *IJCV*, 2004.

[12] S. Mika, G. Rätsch, J. Weston, B. Schölkopf, and K.-R. Müller. Fisher discriminant analysis with kernels. In *Neural Networks for Signal Processing*, 1999.

[13] J. Mutch and D. Lowe. Multiclass object recognition with sparse, localized features. In *CVPR*, 2006.

[14] A. Rakotomamonjy, F. Bach, S. Canu, and Y. Grandvalet. More efficiency in multiple kernel learning. In *ICML*, 2007.

[15] T. Serre, L. Wolf, and T. Poggio. Object recognition with features inspired by visual cortex. In *CVPR*, 2005.

[16] S. Sonnenburg, G. Rätsch, C. Schäfer, and B. Schölkopf. Large scale multiple kernel learning. *JMLR*, 2006.

[17] L. Vandenberghe and S. Boyd. Semidefinite programming. *SIAM Review*, 1996.

[18] M. Varma and D. Ray. Learning the discriminative power-invariance trade-off. In *ICCV*, 2007.

[19] S. Yan, D. Xu, B. Zhang, H. Zhang, Q. Yang, and S. Lin. Graph embedding and extensions: A general framework for dimensionality reduction. *PAMI*, 2007.

[20] H. Zhang, A. Berg, M. Maire, and J. Malik. Svm-knn: Discriminative nearest neighbor classification for visual category recognition. In *CVPR*, 2006.

[21] J. Zhu, S. Rosset, H. Zou, and T. Hastie. Multi-class adaboost. Technical report, Dept. of Statistics, University of Michigan, 2005.
